# Linear Multilayer Independent Component Analysis for Large Natural Scenes

**Yoshitatsu Matsuda** *
Kazunori Yamaguchi Laboratory
Department of General Systems Studies
Graduate School of Arts and Sciences
The University of Tokyo
Japan 153-8902
matsuda@graco.c.u-tokyo.ac.jp

**Kazunori Yamaguchi**
yamaguch@graco.c.u-tokyo.ac.jp

## Abstract

In this paper, linear multilayer ICA (LMICA) is proposed for extracting independent components from quite high-dimensional observed signals such as large-size natural scenes. There are two phases in each layer of LMICA. One is the mapping phase, where a one-dimensional mapping is formed by a stochastic gradient algorithm which makes more highly-correlated (non-independent) signals be nearer incrementally. Another is the local-ICA phase, where each neighbor (namely, highly-correlated) pair of signals in the mapping is separated by the MaxKurt algorithm. Because LMICA separates only the highly-correlated pairs instead of all ones, it can extract independent components quite efficiently from appropriate observed signals. In addition, it is proved that LMICA always converges. Some numerical experiments verify that LMICA is quite efficient and effective in large-size natural image processing.

## 1 Introduction

Independent component analysis (ICA) is a recently-developed method in the fields of signal processing and artificial neural networks, and has been shown to be quite useful for the blind separation problem [1][2][3] [4]. The linear ICA is formalized as follows. Let $s$ and $A$ are $N$-dimensional source signals and $N \times N$ mixing matrix. Then, the observed signals $x$ are defined as

$$x = As. \tag{1}$$

The purpose is to find out $A$ (or the inverse $W$) when the observed (mixed) signals only are given. In other words, ICA blindly extracts the source signals from $M$ samples of the observed signals as follows:

$$\hat{S} = WX, \tag{2}$$

where $\boldsymbol{X}$ is an $N \times M$ matrix of the observed signals and $\hat{\boldsymbol{S}}$ is the estimate of the source signals. This is a typical ill-conditioned problem, but ICA can solve it by assuming that the source signals are generated according to independent and non-gaussian probability distributions. In general, the ICA algorithms find out $\boldsymbol{W}$ by maximizing a criterion (called the contrast function) such as the higher-order statistics (e.g. the kurtosis) of every component of $\hat{\boldsymbol{S}}$. That is, the ICA algorithms can be regarded as an optimization method of such criteria. Some efficient algorithms for this optimization problem have been proposed, for example, the fast ICA algorithm [5][6], the relative gradient algorithm [4], and JADE [7][8].

Now, suppose that quite high-dimensional observed signals (namely, $N$ is quite large) are given such as large-size natural scenes. In this case, even the efficient algorithms are not much useful because they have to find out all the $N^2$ components of $\boldsymbol{W}$. Recently, we proposed a new algorithm for this problem, which can find out global independent components by integrating the local ICA modules. Developing this approach in this paper, we propose a new efficient ICA algorithm named " the linear multilayer ICA algorithm (LMICA)." It will be shown in this paper that LMICA is quite efficient than other standard ICA algorithms in the processing of natural scenes. This paper is an extension of our previous works [9][10].

This paper is organized as follows. In Section 2, the algorithm is described. In Section 3, numerical experiments will verify that LMICA is quite efficient in image processing and can extract some interesting edge detectors from large natural scenes. Lastly, this paper is concluded in Section 4.

## 2   Algorithm

### 2.1   basic idea

LMICA can extract all the independent components approximately by repetition of the following two phases. One is the mapping phase, which brings more highly-correlated signals nearer. Another is local-ICA phase, where each neighbor pair of signals in the mapping is separated by MaxKurt algorithm [8]. The mechanism of LMICA is illustrated in Fig. 1. Note that this illustration holds just in the ideal case where the mixing matrix $\boldsymbol{A}$ is given according to such a hierarchical model. In other words, it *does not* hold for an arbitrary $\boldsymbol{A}$. It will be shown in Section 3 that this hierarchical model is quite effective at least in natural scenes.

### 2.2   mapping phase

In the mapping phase, given signals $\boldsymbol{X}$ are arranged in a one-dimensional array so that pairs $(i, j)$ taking higher $\sum_k x_{ik}^2 x_{jk}^2$ are placed nearer. Letting $\boldsymbol{Y} = (y_i)$ be the coordinate of the $i$-th signal $x_{ik}$, the following objective function $\mu$ is defined:

$$\mu\left(\boldsymbol{Y}\right) = \sum_{i,j} \sum_k x_{ik}^2 x_{jk}^2 \left(y_i - y_j\right)^2. \tag{3}$$

The optimal mapping is found out by minimizing $\mu$ with respect to $\boldsymbol{Y}$ under the constraints that $\sum y_i = 0$ and $\sum y_i^2 = 1$. It has been well-known that such optimization problems can be solved efficiently by a stochastic gradient algorithm [11][12]. In this case, the stochastic gradient algorithm is given as follows (see [10] for the details of the derivation of this algorithm):

$$y_i\left(T+1\right) := y_i\left(T\right) - \lambda_T \left(z_i y_i \zeta + z_i \eta\right), \tag{4}$$

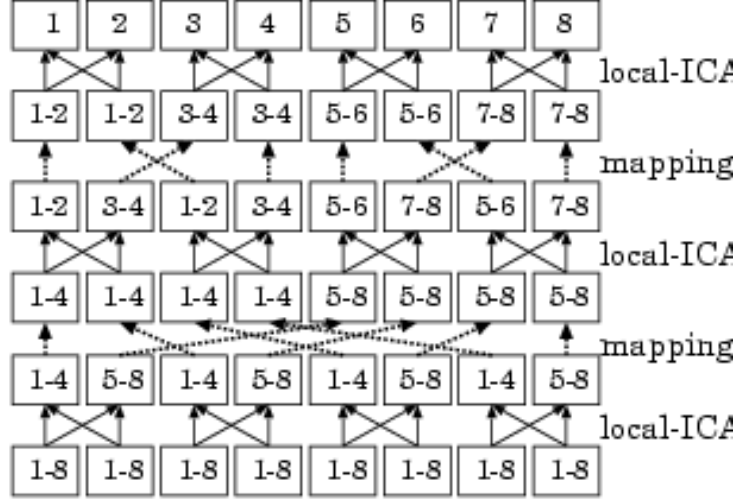

Figure 1: The illustration of LMICA (the ideal case): Each number from 1 to 8 means a source signal. In the first local-ICA phase, each neighbor pair of the completely-mixed signals (denoted "1-8") is partially separated into "1-4" and "5-8." Next, the mapping phase rearranges the partially-separated signals so that more highly-correlated signals are nearer. In consequence, the four "1-4" signals (similarly, "5-8" ones) are brought nearer. Then, the local-ICA phase partially separates the pairs of neighbor signals into "1-2," "3-4," "5-6," and "7-8." By repetition of the two phases, LMICA can extract all the sources quite efficiently.

where $\lambda_T$ is the step size at the $T$-th time step, $z_i = x_{ik}^2$ ($k$ is randomly selected from $\{1, \ldots, M\}$ at each time step),

$$\zeta = \sum_i z_i, \tag{5}$$

and

$$\eta = \sum_i z_i y_i. \tag{6}$$

By calculating $\zeta$ and $\eta$ before the update for each $i$, each update requires just $O(N)$ computation. Eq. (4) is guaranteed to converge to a local minimum of the objective function $\mu(\boldsymbol{Y})$ if $\lambda_T$ decreases sufficiently slowly ($\lim_{T \to \infty} \lambda_T = 0$ and $\sum \lambda_T = \infty$).

Because the $\boldsymbol{Y}$ in the above method is continuous, each continuous $y_i$ is replaced by the ranking of itself in $\boldsymbol{Y}$ in the last of the mapping phase. That is, $y_i := 1$ for the largest $y_i$, $y_j := N$ for the smallest one, and so on. The corresponding permutation $\sigma$ is given as $\sigma(i) = y_i$.

The total procedure of the mapping phase for given $\boldsymbol{X}$ is described as follows:

---

*mapping phase*

1. $x_{ik} := x_{ik} - \bar{x}_i$ for each $i, k$, where $\bar{x}_i$ is the mean $\frac{\sum_k x_{ik}}{M}$.

2. $y_i = i$, and $\sigma(i) = i$ for each $i$.

3. Until the convergence, repeat the following steps:

    (a) Select $k$ randomly from $\{1, \ldots, M\}$, and let $z_i = x_{ik}^2$ for each $i$.

    (b) Update each $y_i$ by Eq. (4).

    (c) Normalize $\boldsymbol{Y}$ to satisfy $\sum_i y_i = 0$ and $\sum_i y_i^2 = 1$.

4. Discretize $y_i$.

5. Update $\boldsymbol{X}$ by $x_{\sigma(i)k} := x_{ik}$ for each $i$ and $k$.

---

## 2.3 local-ICA phase

In the local-ICA phase, the following contrast function $\phi(\boldsymbol{X})$ (the sum of kurtoses) is used (MaxKurt algorithm in [8]):

$$\phi(\boldsymbol{X}) = -\sum_{i,k} x_{ik}^4, \tag{7}$$

and $\phi(\boldsymbol{X})$ is minimized by "rotating" the neighbor pairs of signals (namely, under an orthogonal transformation). For each neighbor pair $(i, i+1)$, a rotation matrix $\boldsymbol{R}_i(\theta)$ is given as

$$\boldsymbol{R}_i(\theta) = \begin{pmatrix} \boldsymbol{I}_{i-1} & \boldsymbol{0} & \boldsymbol{0} & \boldsymbol{0} \\ \boldsymbol{0} & \cos\theta & \sin\theta & \boldsymbol{0} \\ \boldsymbol{0} & -\sin\theta & \cos\theta & \boldsymbol{0} \\ \boldsymbol{0} & \boldsymbol{0} & \boldsymbol{0} & \boldsymbol{I}_{N-i-2} \end{pmatrix}, \tag{8}$$

where $\boldsymbol{I}_n$ is the $n \times n$ identity matrix. Then, the optimal angle $\hat{\theta}$ is given as

$$\hat{\theta} = \mathrm{argmin}_\theta \phi(\boldsymbol{X}'), \tag{9}$$

where $\boldsymbol{X}'(\theta) = \boldsymbol{R}_i(\theta)\boldsymbol{X}$. After some tedious transformation of the equations (see [8]), it is shown that $\hat{\theta}$ is determined analytically by the following equations:

$$\sin 4\hat{\theta} = \frac{\alpha_{ij}}{\sqrt{\alpha_{ij}^2 + \beta_{ij}^2}}, \cos 4\hat{\theta} = \frac{\beta_{ij}}{\sqrt{\alpha_{ij}^2 + \beta_{ij}^2}}, \tag{10}$$

where

$$\alpha_{ij} = \sum_k \left( x_{ik}^3 x_{jk} - x_{ik} x_{jk}^3 \right), \beta_{ij} = \frac{\sum_k \left( x_{ik}^4 + x_{jk}^4 - 6x_{ik}^2 x_{jk}^2 \right)}{4}, \tag{11}$$

and $j = i + 1$.

Now, the procedure of the local-ICA phase for given $\boldsymbol{X}$ is described as follows:

---

*local-ICA phase*

1. Let $\boldsymbol{W}_{\mathrm{local}} = \boldsymbol{I}_N$, $\boldsymbol{A}_{\mathrm{local}} = \boldsymbol{I}_N$

2. For each $i = \{1, \ldots, N-1\}$,

    (a) Find out the optimal angle $\hat{\theta}$ by Eq. (10).

    (b) $\boldsymbol{X} := \boldsymbol{R}_i(\hat{\theta})\boldsymbol{X}$, $\boldsymbol{W}_{\mathrm{local}} := \boldsymbol{R}_i \boldsymbol{W}_{\mathrm{local}}$, and $\boldsymbol{A}_{\mathrm{local}} := \boldsymbol{A}_{\mathrm{local}} \boldsymbol{R}_i^t$.

---

### 2.4 complete algorithm

The complete algorithm of LMICA for any given observed signals $\boldsymbol{X}$ is given by repeating the mapping phase and the local-ICA phase alternately. Here, $\boldsymbol{P}_\sigma$ is the permutation matrix corresponding to $\sigma$.

---

*linear multilayer ICA algorithm*

1. Initial Settings: Let $\boldsymbol{X}$ be the given observed signal matrix, and $\boldsymbol{W}$ and $\boldsymbol{A}$ be $\boldsymbol{I}_N$.

2. Repetition: Do the following two phases alternately over $L$ times.

   (a) Mapping Phase: Find out the optimal permutation matrix $\boldsymbol{P}_\sigma$ and the optimally-arranged signals $\boldsymbol{X}$ by the mapping phase. Then, $\boldsymbol{W} := \boldsymbol{P}_\sigma \boldsymbol{W}$ and $\boldsymbol{A} := \boldsymbol{A}\boldsymbol{P}_\sigma^t$.

   (b) Local-ICA Phase: Find out the optimal matrices $\boldsymbol{W}_{\text{local}}$, $\boldsymbol{A}_{\text{local}}$, and $\boldsymbol{X}$. Then, $\boldsymbol{W} := \boldsymbol{W}_{\text{local}}\boldsymbol{W}$ and $\boldsymbol{A} := \boldsymbol{A}\boldsymbol{A}_{\text{local}}$.

---

### 2.5 some remarks

**Relation to MaxKurt algorithm.** Eq. (10) is just the same as MaxKurt algorithm [8]. The crucial difference between our LMICA and MaxKurt is that LMICA optimizes just the neighbor pairs instead of all the $\frac{N(N-1)}{2}$ ones in MaxKurt. In LMICA, the pairs with higher "costs" (higher $\sum_k x_{ik}^2 x_{jk}^2$) are brought nearer in the mapping phase. So, independent components can be extracted effectively by optimizing just the neighbor pairs.

**Contrast function.** In order to make consistency between this paper and our previous work [10], the following contrast function $\phi$ instead of Eq. (7) is used in Section 3:

$$\phi(\boldsymbol{X}) = \sum_{i,j,k} x_{ik}^2 x_{jk}^2. \tag{12}$$

The minimization of Eq. (12) is equivalent to that of Eq. (7) under the orthogonal transformation.

**Pre-whitening.** Though LMICA (which is based on MaxKurt) presupposes that $\boldsymbol{X}$ is pre-whitened, the algorithm in Section 2.4 is applicable to any raw $\boldsymbol{X}$ without the pre-whitening. Because any pre-whitening method suitable for LMICA has not been found out yet, raw images of natural scenes are given as $\boldsymbol{X}$ in the numerical experiments in Section 3. In this non-whitening case, the mixing matrix $\boldsymbol{A}$ is limited to be orthogonal and the influence of the second-order statistics is not removed. Nevertheless, it will be shown in Section 3 that the higher-order statistics of $\boldsymbol{X}$ cause some interesting results.

## 3  Results

It has been well-known that various local edge detectors can be extracted from natural scenes by the standard ICA algorithm [13][14]. Here, LMICA was applied to the same problem. 30000 samples of natural scenes of $12 \times 12$ pixels were given as the observed signals $\boldsymbol{X}$. That is, $N$ and $M$ were 144 and 30000. Original natural scenes were downloaded at `http://www.cis.hut.fi/projects/ica/data/images/`. The number of

layers $L$ was set 720, where one layer means one pair of the mapping and the local-ICA phases. For comparison, the experiments without the mapping phase were carried out, where the mapping $Y$ was randomly generated. In addition, the standard MaxKurt algorithm [8] was used with 10 iterations. The contrast function $\phi$ (Eq. (12)) was calculated at each layer, and it was averaged over 10 independently generated $X$s. Fig. 2-(a) shows the decreasing curves of $\phi$ of normal LMICA and the one without the mapping phase. The cross points show the result at each iteration of MaxKurt. Because one iteration of MaxKurt is equivalent to 72 layers of LMICA with respect to the times of the optimizations for the pairs of signals, a scaling ($\times 72$) is applied. Surprisingly, LMICA nearly converged to the optimal point within just 10 layers. The number of parameters within 10 layers is $143 \times 10$, which is much fewer than the degree of freedom of $A$ ($\frac{144 \times 143}{2}$). It suggests that LMICA gives a quite suitable model for natural scenes. The calculation time with the values of $\phi$ is shown in Table. 1. It shows that the time costs of the mapping phase are not much higher than those of the local-ICA phase. The fact that 10 layers of LMICA required much less time (22sec.) than one iteration of MaxKurt (94sec.) and optimized $\phi$ approximately (4.91) verifies the efficiency of LMICA. Note that each iteration of MaxKurt can not be stopped halfway. Fig. 3 shows $5 \times 5$ representative edge detectors at each layer of LMICA. At the 20th layer (Fig. 3-(a)), rough and local edge detectors were recognized, though they were a little unclear. As the layer proceeded, edge detectors became clearer and more global (see Figs. 3-(b) and 3-(c)). It is interesting that ICA-like local edges (where the higher-order statistics are dominant) at the early stage were transformed to PCA-like global edges (the second-order statistics are dominant) at the later stage (see [13]). For comparison, Fig. 3-(d) show the result at the 10th iteration of MaxKurt. It is similar to Fig. 3-(c) as expected.

In addition, we used large-size natural scenes. 100000 samples of natural scenes of $64 \times 64$ pixels were given as $X$. MaxKurt and other well-known ICA algorithms are not available for such a large-scale problem because they require huge computation. Fig. 2-(b) shows the decreasing curve of $\phi$ in the large-size natural scenes. LMICA was carried out in 1000 layers, and it consumed about 69 hours with Intel 2.8GHz CPU. It shows that LMICA rapidly decreased in the first 20 layers and converged around the 500th layer. It verifies that LMICA is quite efficient in the analysis of large-size natural scenes. Fig. 4 shows some edge detectors generated at the 1000th layer. It is interesting that some "compound" detectors such as a "cross" were generated in addition to simple "long-edge" detectors. In a famous previous work [13] which applied ICA and PCA to small-size natural scenes, symmetric global edge detectors similar to our "compound" ones could be generated by PCA which manages only the second-order statistics. On the other hand, asymmetric local edge detectors similar to our simple "long-edge" ones could not be generated by PCA and could be extracted by ICA utilizing the higher-order statistics. In comparison with it, our LMICA could extract various local and global detectors simultaneously from large-size natural scenes. Besides, it is expected from the results for small-size images (see Fig. 3) that other various detectors are generated at each layer. In summary, those results show that LMICA can extract quite many useful and various detectors from large-size natural scenes efficiently. It is also interesting that there was a plateau in the neighborhood of the 10th layer. It suggests that large-size natural scenes may be generated by two different generative models. But, the close inspection is beyond the scope of this paper.

## 4 Conclusion

In this paper, we proposed the linear multilayer ICA algorithm (LMICA). We carried out some numerical experiments on natural scenes, which verified that LMICA can find out the approximations of independent components quite efficiently and it is applicable to large problems. We are now analyzing the results of LMICA in large-size natural scenes of $64 \times 64$ pixels, and we are planning to apply this algorithm to quite large-scale images such as the ones of $256 \times 256$ pixels. We are also planning to utilize LMICA in the data mining

Table 1: Calculation time with the values of the contrast function $\phi$ (Eq. (12)): They are the averages over 10 runs at the 10th layer (approximation) and the 720th layer (convergence) in LMICA (the normal one and the one without the mapping phase). In addition, those of 10 iterations in MaxKurt (approximately corresponding to $L = 10 \times 72 = 720$) are shown. They were calculated in Intel 2.8GHz CPU.

|  | LMICA | LMICA without mapping | MaxKurt (10 iterations) |
|---|---|---|---|
| 10th layer | 22sec. (4.91) | 9.3sec. (17.6) | - |
| 720th layer | 1600sec. (4.57) | 670sec. (4.57) | 940sec. (4.57) |

of quite high-dimensional data space, such as the text mining. In addition, we are trying to find out the pre-whitening method suitable for LMICA. Some normalization techniques in the local-ICA phase may be promising.

## Footnotes

*http://www.graco.c.u-tokyo.ac.jp/~matsuda

# References

[1] C. Jutten and J. Herault. Blind separation of sources (part I): An adaptive algorithm based on neuromimetic architecture. *Signal Processing*, 24(1):1–10, jul 1991.

[2] P. Comon. Independent component analysis - a new concept? *Signal Processing*, 36:287–314, 1994.

[3] A. J. Bell and T. J. Sejnowski. An information-maximization approach to blind separation and blind deconvolution. *Neural Computation*, 7:1129–1159, 1995.

[4] J.-F. Cardoso and Beate Laheld. Equivariant adaptive source separation. *IEEE Transactions on Signal Processing*, 44(12):3017–3030, dec 1996.

[5] A. Hyvärinen and E. Oja. A fast fixed-point algorithm for independent component analysis. *Neural Computation*, 9(7):1483–1492, 1997.

[6] A. Hyvärinen. Fast and robust fixed-point algorithms for independent component analysis. *IEEE Transactions on Neural Networks*, 10(3):626–634, 1999.

[7] Jean-François Cardoso and Antoine Souloumiac. Blind beamforming for non Gaussian signals. *IEE Proceedings-F*, 140(6):362–370, dec 1993.

[8] Jean-François Cardoso. High-order contrasts for independent component analysis. *Neural Computation*, 11(1):157–192, jan 1999.

[9] Yoshitatsu Matsuda and Kazunori Yamaguchi. Linear multilayer ica algorithm integrating small local modules. In *Proceedings of ICA2003*, pages 403–408, Nara, Japan, 2003.

[10] Yoshitatsu Matsuda and Kazunori Yamaguchi. Linear multilayer independent component analysis using stochastic gradient algorithm. In *Independent Component Analysis and Blind source separation - ICA2004*, volume 3195 of *LNCS*, pages 303–310, Granada, Spain, sep 2004. Springer-Verlag.

[11] Yoshitatsu Matsuda and Kazunori Yamaguchi. Global mapping analysis: stochastic approximation for multidimensional scaling. *International Journal of Neural Systems*, 11(5):419–426, 2001.

[12] Yoshitatsu Matsuda and Kazunori Yamaguchi. An efficient MDS-based topographic mapping algorithm. *Neurocomputing*, 2005. in press.

[13] A. J. Bell and T. J. Sejnowski. The "independent components" of natural scenes are edge filters. *Vision Research*, 37(23):3327–3338, dec 1997.

[14] J. H. van Hateren and A. van der Schaaf. Independent component filters of natural images compared with simple cells in primary visual cortex. *Proceedings of the Royal Society of London: B*, 265:359–366, 1998.

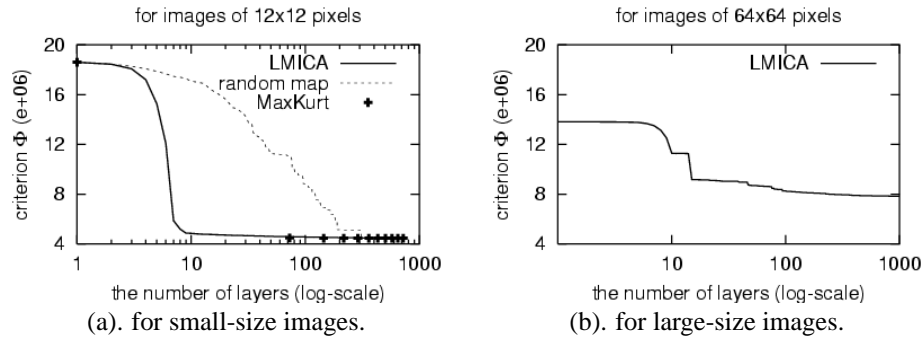

for images of 12x12 pixels      for images of 64x64 pixels

(a). for small-size images.     (b). for large-size images.

Figure 2: Decreasing curve of the contrast function $\phi$ along the number of layers (in log-scale): (a). It is for small-size natural scenes of $12 \times 12$ pixels. The normal and dotted curves show the decreases of $\phi$ by LMICA and the one without the mapping phase (random mapping), respectively. The cross points show the results of MaxKurt. Each iteration in MaxKurt approximately corresponds to 72 layers with respect to the times of the optimizations for the pairs of signals. (b). It is for large-size natural scenes of $64 \times 64$ pixels. The curve displays the decrease of $\phi$ by LMICA in 1000 layers.

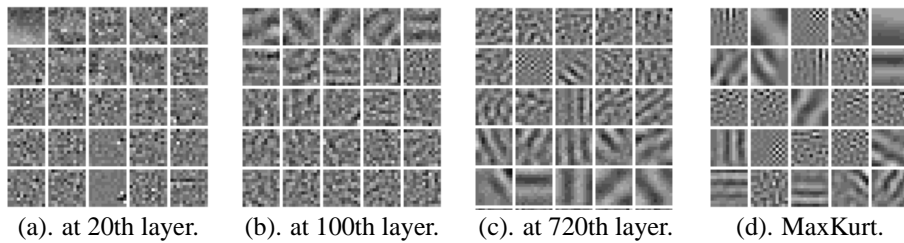

(a). at 20th layer.    (b). at 100th layer.    (c). at 720th layer.    (d). MaxKurt.

Figure 3: Representative edge detectors from natural scenes of $12 \times 12$ pixels: (a). It displays the basis vectors generated by LMICA at the 20th layer. (b). at the 100th layer. (c). at the 720th layer. (d). It shows the ones after 10 iterations of MaxKurt algorithm.

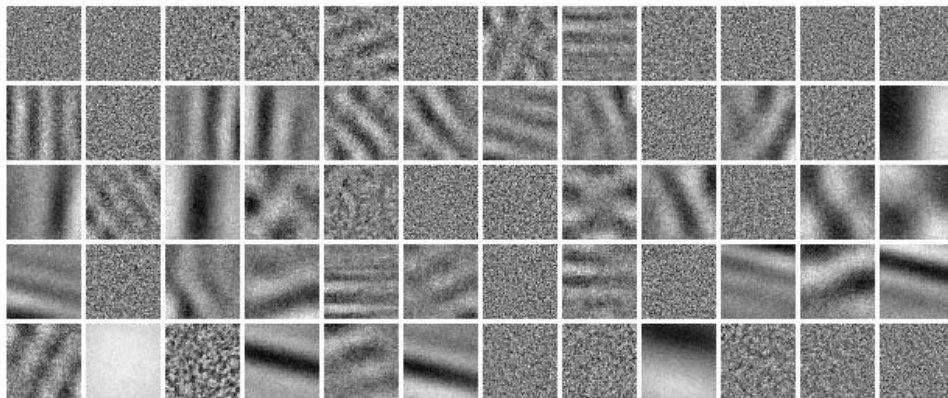

Figure 4: Representative edge detectors from natural scenes of $64 \times 64$ pixels.